# Combining Graph Laplacians for Semi–Supervised Learning

**Andreas Argyriou,    Mark Herbster,    Massimiliano Pontil**

Department of Computer Science
University College London
Gower Street, London WC1E 6BT, England, UK
*{a.argyriou, m.herbster, m.pontil}@cs.ucl.ac.uk*

## Abstract

A foundational problem in semi-supervised learning is the construction of a graph underlying the data. We propose to use a method which optimally combines a number of differently constructed graphs. For each of these graphs we associate a basic graph kernel. We then compute an optimal *combined* kernel. This kernel solves an extended regularization problem which requires a joint minimization over both the data and the set of graph kernels. We present encouraging results on different OCR tasks where the optimal combined kernel is computed from graphs constructed with a variety of distances functions and the '$k$' in nearest neighbors.

## 1   Introduction

Semi-supervised learning has received significant attention in machine learning in recent years, see, for example, [2, 3, 4, 8, 9, 16, 17, 18] and references therein. The defining insight of semi-supervised methods is that unlabeled data may be used to improve the performance of learners in a supervised task. One of the key semi-supervised learning methods builds on the assumption that the data is situated on a low dimensional manifold within the ambient space of the data and that this manifold can be approximated by a weighted discrete graph whose vertices are identified with the empirical (labeled and unlabeled) data, [3, 17]. Graph construction consists of two stages, first selection of a distance function and then application of it to determine the graph's edges (or weights thereof). For example, in this paper we consider distances between images based on the Euclidean distance, Euclidean distance combined with image transformations, and the related tangent distance [6]; we determine the edge set of the graph with $k$-nearest neighbors. Another common choice is to weight edges by a decreasing function of the distance $d$ such as $e^{-\beta d^2}$.

Although a surplus of unlabeled data may improve the quality of the empirical approximation of the manifold (via the graph) leading to improved performances, practical experience with these methods indicates that their performance significantly depends on how the graph is constructed. Hence, the model selection problem must consider both the selection of the distance function and the parameters $k$ or $\beta$ used in the graph building process described above. A diversity of methods have been proposed for graph construction; in this paper

we do not advocate *selecting* a single graph but, rather we propose *combining* a number of graphs. Our solution implements a method based on regularization which builds upon the work in [1]. For a given dataset each combination of distance functions and edge set specifications from the distance will lead to a specific graph. Each of these graphs may then be associated with a kernel. We then apply regularization to select the best convex combination of these kernels; the minimizing function will trade off its fit to the data against its norm. What is unique about this regularization is that the minimization is not over a single kernel space but rather over a space corresponding to all convex combinations of kernels. Thus all data (labeled vertices) may be conserved for training rather than reduced by cross-validation which is not an appealing option when the number of labeled vertices per class is very small.

Figure 3 in Section 4 illustrates our algorithm on a simple example. There, three different distances for 400 images of the digits 'six' and 'nine' are depicted, namely, the Euclidean distance, a distance invariant under small centered image rotations from $[-10°, 10°]$ and a distance invariant under rotations from $[-180°, 180°]$. Clearly, the last distance is problematic as sixes become similar to nines. The performance of our graph regularization learning algorithm discussed in Section 2.2 with these distances is reported below each plot; as expected, this performance is much lower in the case that the third distance is used. The paper is constructed as follows. In Section 2 we discuss how regularization may be applied to single graphs. First, we review regularization in the context of reproducing kernel Hilbert spaces (Section 2.1); then in Section 2.2 we specialize our discussion to Hilbert spaces of functions defined over a graph. Here we review the (normalized) Laplacian of the graph and a kernel which is the pseudoinverse of the graph Laplacian. In Section 3 we detail our algorithm for learning an optimal convex combination of Laplacian kernels. Finally, in Section 4 we present experiments on the USPS dataset with our algorithm trained over different classes of Laplacian kernels.

## 2 Background on graph regularization

In this section we review graph regularization [2, 9, 14] from the perspective of reproducing kernel Hilbert spaces, see e.g. [12].

### 2.1 Reproducing kernel Hilbert spaces

Let $X$ be a set and $K : X \times X \to \mathbb{R}$ a kernel function. We say that $\mathcal{H}_K$ is a reproducing kernel Hilbert space (RKHS) of functions $f : X \to \mathbb{R}$ if (i): for every $x \in X$, $K(x, \cdot) \in \mathcal{H}_K$ and (ii): the *reproducing kernel property* $f(x) = \langle f, K(x, \cdot) \rangle_K$ holds for every $f \in \mathcal{H}_K$ and $x \in X$, where $\langle \cdot, \cdot \rangle_K$ is the inner product on $\mathcal{H}_K$. In particular, (ii) tells us that for $x, t \in X$, $K(x, t) = \langle K(x, \cdot), K(t, \cdot) \rangle_K$, implying that the $n \times n$ matrix $(K(t_i, t_j) : i, j \in \mathbb{N}_p)$ is symmetric and positive semi-definite for *any* set of inputs $\{t_i : i \in \mathbb{N}_p\} \subseteq X$, $p \in \mathbb{N}$, where we use the notation $\mathbb{N}_p := \{1, \ldots, p\}$.

Regularization in an RKHS learns a function $f \in \mathcal{H}_K$ on the basis of available input/output examples $\{(x_i, y_i) : i \in \mathbb{N}_\ell\}$ by solving the variational problem

$$E_\gamma(K) := \min \left\{ \sum_{i=1}^{\ell} V(y_i, f(x_i)) + \gamma \|f\|_K^2 : f \in \mathcal{H}_K \right\} \qquad (2.1)$$

where $V : \mathbb{R} \times \mathbb{R} \to [0, \infty)$ is a loss function and $\gamma$ a positive parameter. Moreover, if $f$ is a solution to problem (2.1) then it has the form

$$f(x) = \sum_{i=1}^{\ell} c_i K(x_i, x), \quad x \in X \qquad (2.2)$$

for some real vector of coefficients $\mathbf{c} = (c_i : i \in \mathbb{N}_\ell)^\top$, see, for example, [12], where "$\top$" denotes transposition. This vector can be found by replacing $f$ by the right hand side of equation (2.2) in equation (2.1) and then optimizing with respect to $\mathbf{c}$. However, in many practical situations it is more convenient to compute $\mathbf{c}$ by solving the dual problem to (2.1), namely

$$-E_\gamma(K) := \min\left\{ \frac{1}{4\gamma}\mathbf{c}^\top \widetilde{\mathbf{K}}\mathbf{c} + \sum_{i=1}^{\ell} V^*(y_i, c_i) : \mathbf{c} \in \mathbb{R}^\ell \right\} \tag{2.3}$$

where $\widetilde{\mathbf{K}} = (K(x_i, x_j))_{i,j=1}^\ell$ and the function $V^* : \mathbb{R} \times \mathbb{R} \to \mathbb{R} \cup \{+\infty\}$ is the conjugate of the loss function $V$ which is defined, for every $z, \alpha \in \mathbb{R}$, as $V^*(z, \alpha) := \sup\{\lambda\alpha - V(z, \lambda) : \lambda \in \mathbb{R}\}$, see, for example, [1] for a discussion. The choice of the loss function $V$ leads to different learning methods among which the most prominent are square loss regularization and support vector machines, see, for example [15].

## 2.2 Graph regularization

Let $G$ be an undirected graph with $m$ vertices and an $m \times m$ adjacency matrix $\mathbf{A}$ such that $A_{ij} = 1$ if there is an edge connecting vertices $i$ and $j$ and zero otherwise[1]. The graph Laplacian $\mathbf{L}$ is the $m \times m$ matrix defined as $\mathbf{L} := \mathbf{D} - \mathbf{A}$, where $\mathbf{D} = \mathrm{diag}(d_i : i \in \mathbb{N}_m)$ and $d_i$ is the degree of vertex $i$, that is $d_i = \sum_{j=1}^m A_{ij}$.

We identify the linear space of real-valued functions defined on the graph with $\mathbb{R}^m$ and introduce on it the semi-inner product

$$\langle \mathbf{u}, \mathbf{v} \rangle := \mathbf{u}^\top \mathbf{L}\mathbf{v}, \quad \mathbf{u}, \mathbf{v} \in \mathbb{R}^m.$$

The induced semi-norm is $\|\mathbf{v}\| := \sqrt{\langle \mathbf{v}, \mathbf{v} \rangle}$, $\mathbf{v} \in \mathbb{R}^m$. It is a semi-norm since $\|\mathbf{v}\| = 0$ if $\mathbf{v}$ is a constant vector, as can be verified by noting that $\|\mathbf{v}\|^2 = \frac{1}{2}\sum_{i,j=1}^m (v_i - v_j)^2 A_{ij}$.

We recall that $G$ has $r$ connected components if and only if $\mathbf{L}$ has $r$ eigenvectors with zero eigenvalues. Those eigenvectors are piece-wise constant on the connected components of the graph. In particular, $G$ is connected if and only if the constant vector is the only eigenvector of $\mathbf{L}$ with zero eigenvalue [5]. We let $\{\sigma_i, \mathbf{u}_i\}_{i=1}^m$ be a system of eigenvalues/vectors of $\mathbf{L}$ where the eigenvalues are non-decreasing in order, $\sigma_i = 0$, $i \in \mathbb{N}_r$, and define the linear subspace $\mathcal{H}(G)$ of $\mathbb{R}^m$ which is orthogonal to the eigenvectors with zero eigenvalue, that is,

$$\mathcal{H}(G) := \{\mathbf{v} : \mathbf{v}^\top \mathbf{u}_i = 0, \ i \in \mathbb{N}_r\}.$$

Within this framework, we wish to learn a function $\mathbf{v} \in \mathcal{H}(G)$ on the basis of a set of labeled vertices. Without loss of generality we assume that the first $\ell \leq m$ vertices are labeled and let $y_1, ..., y_\ell \in \{-1, 1\}$ be the corresponding labels. Following [2] we prescribe a loss function $V$ and compute the function $\mathbf{v}$ by solving the optimization problem

$$\min\left\{ \sum_{i=1}^{\ell} V(y_i, v_i) + \gamma\|\mathbf{v}\|^2 : \mathbf{v} \in \mathcal{H}(G) \right\}. \tag{2.4}$$

We note that a similar approach is presented in [17] where $\mathbf{v}$ is (essentially) obtained as the minimal norm interpolant in $\mathcal{H}(G)$ to the labeled vertices. The functional (2.4) balances the error on the labeled points with a smoothness term measuring the complexity of $\mathbf{v}$ on the graph. Note that this last term contains the information of both the labeled and unlabeled vertices via the graph Laplacian.

Method (2.4) is a special case of problem (2.1). Indeed, the restriction of the semi-norm $\|\cdot\|$ on $\mathcal{H}(G)$ is a norm. Moreover, the pseudoinverse of the Laplacian, $\mathbf{L}^+$, is the reproducing kernel of $\mathcal{H}(G)$, see, for example, [7] for a proof. This means that for every $\mathbf{v} \in \mathcal{H}(G)$ and $i \in \mathbb{N}_m$ there holds the reproducing kernel property $v_i = \langle \mathbf{L}_i^+, \mathbf{v} \rangle$, where $\mathbf{L}_i^+$ is the $i$-th column of $\mathbf{L}^+$. Hence, by setting $X \equiv \mathbb{N}_m$, $f(i) = v_i$ and $K(i,j) = L_{ij}^+$, $i, j \in \mathbb{N}_m$, we see that $\mathcal{H}_K \equiv \mathcal{H}(G)$. We note that the above analysis naturally extends to the case that $\mathbf{L}$ is replaced by any positive semidefinite matrix. In particular, in our experiments below we will use the normalized Laplacian matrix given by $\mathbf{D}^{-\frac{1}{2}} \mathbf{L} \mathbf{D}^{-\frac{1}{2}}$.

Typically, problem (2.4) is solved by optimizing over $\mathbf{v} = (v_i : i \in \mathbb{N}_m)$. In particular, for square loss regularization [2] and minimal norm interpolation [17] this requires solving a squared linear system of $m$ and $m - \ell$ equations respectively. On the contrary, in this paper we use the representer theorem to express $\mathbf{v}$ as

$$\mathbf{v} = \Big( \sum_{j=1}^{\ell} L_{ij}^+ c_j : i \in \mathbb{N}_m \Big).$$

This approach is advantageous if $\mathbf{L}^+$ can be computed off-line because, typically, $\ell \ll m$. A further advantage of this approach is that multiple problems may be solved with the same Laplacian kernel. The coefficients $c_i$ are obtained by solving problem (2.3) with $\widetilde{\mathbf{K}} = (L_{ij}^+)_{i,j=1}^{\ell}$. For example, for square loss regularization the computation of the parameter vector $\mathbf{c} = (c_i : i \in \mathbb{N}_\ell)$ involves solving a linear system of $\ell$ equations, namely

$$(\widetilde{\mathbf{K}} + \gamma \mathbf{I})\mathbf{c} = \mathbf{y}. \tag{2.5}$$

## 3 Learning a convex combination of Laplacian kernels

We now describe our framework for learning with multiple graph Laplacians. We assume that we are given $n$ graphs $G^{(q)}$, $q \in \mathbb{N}_n$, all having $m$ vertices, with corresponding Laplacians $\mathbf{L}^{(q)}$, kernels $K^{(q)} = (\mathbf{L}^{(q)})^+$, Hilbert spaces $\mathcal{H}^{(q)} := \mathcal{H}(G^{(q)})$ and norms $\|\mathbf{v}\|_q^2 := \mathbf{v}^\top \mathbf{L}^{(q)} \mathbf{v}$, $\mathbf{v} \in \mathcal{H}^{(q)}$. We propose to learn an optimal convex combination of graph kernels, that is, we solve the optimization problem

$$\rho = \min \left\{ \sum_{i=1}^{\ell} V(y_i, v_i) + \gamma \|\mathbf{v}\|_{K(\boldsymbol{\lambda})}^2 : \boldsymbol{\lambda} \in \Lambda, \ \mathbf{v} \in \mathcal{H}_{K(\boldsymbol{\lambda})} \right\} \tag{3.1}$$

where we have defined the set $\Lambda := \{\boldsymbol{\lambda} \in \mathbb{R}^n : \lambda_q \geq 0, \sum_{q=1}^{n} \lambda_q = 1\}$ and, for each $\boldsymbol{\lambda} \in \Lambda$, the kernel $K(\boldsymbol{\lambda}) := \sum_{q=1}^{n} \lambda_q K^{(q)}$. The above problem is motivated by observing that

$$\rho \leq \min \left\{ E_\gamma(K^{(q)}) : q \in \mathbb{N}_n \right\}.$$

Hence an optimal convex combination of kernels has a smaller right hand side than that of any individual kernel, motivating the expectation of improved performance. Furthermore, large values of the components of the minimizing $\boldsymbol{\lambda}$ identify the most relevant kernels.

Problem (3.1) is a special case of the problem of *jointly* minimizing functional (2.1) over $\mathbf{v} \in \mathcal{H}_K$ and $K \in co(\mathcal{K})$, the convex hull of kernels in a prescribed set $\mathcal{K}$. This problem is discussed in detail in [1, 12], see also [10, 11] where the case that $\mathcal{K}$ is finite is considered. Practical experience with this method [1, 10, 11] indicates that it can enhance the performance of the learning algorithm and, moreover, it is computationally efficient to solve. When solving problem (3.1) it is important to require that the kernels $K^{(q)}$ satisfy a normalization condition such as that they all have the same trace or the same Frobenius norm, see [10] for a discussion.

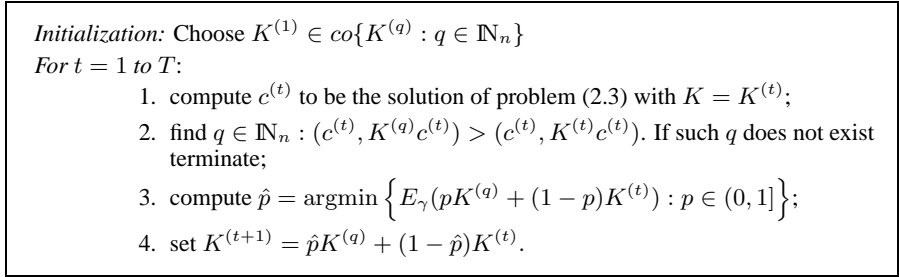

*Initialization:* Choose $K^{(1)} \in co\{K^{(q)} : q \in \mathbb{N}_n\}$

*For* $t = 1$ *to* $T$:

    1. compute $c^{(t)}$ to be the solution of problem (2.3) with $K = K^{(t)}$;

    2. find $q \in \mathbb{N}_n : (c^{(t)}, K^{(q)}c^{(t)}) > (c^{(t)}, K^{(t)}c^{(t)})$. If such $q$ does not exist terminate;

    3. compute $\hat{p} = \text{argmin} \left\{ E_\gamma(pK^{(q)} + (1-p)K^{(t)}) : p \in (0,1] \right\}$;

    4. set $K^{(t+1)} = \hat{p}K^{(q)} + (1 - \hat{p})K^{(t)}$.

Figure 1: Algorithm to compute an optimal convex combination of kernels in the set $co\{K^{(q)} : q \in \mathbb{N}_n\}$.

Using the dual problem formulation discussed above (see equation (2.3)) in the inner minimum in (3.1) we can rewrite this problem as

$$-\rho = \max \left\{ \min \left\{ \frac{1}{4\gamma} \mathbf{c}^\top \widetilde{\mathbf{K}}(\boldsymbol{\lambda})\mathbf{c} + \sum_{i=1}^{\ell} V^*(y_i, c_i) : \mathbf{c} \in \mathbb{R}^\ell \right\} : \boldsymbol{\lambda} \in \Lambda \right\}. \qquad (3.2)$$

The variational problem (3.2) expresses the optimal convex combination of the kernels as the solution to a saddle point problem. This problem is simpler to solve than the original problem (3.1) since its objective function is linear in $\boldsymbol{\lambda}$, see [1] for a discussion. Several algorithms can be used for computing a saddle point $(\hat{\mathbf{c}}, \hat{\boldsymbol{\lambda}}) \in \mathbb{R}^\ell \times \Lambda$. Here we adapt an algorithm from [1] which alternately optimizes over $\mathbf{c}$ and $\boldsymbol{\lambda}$. For reproducibility of the algorithm, it is reported in Figure 1. Note that once $\hat{\boldsymbol{\lambda}}$ is computed $\hat{\mathbf{c}}$ is given by a minimizer of problem (2.3) for $K = K(\boldsymbol{\lambda})$. In particular, for square loss regularization this requires solving the equation (2.5) with $\widetilde{\mathbf{K}} = (K_{ij}(\hat{\boldsymbol{\lambda}}) : i, j \in \mathbb{N}_\ell)$.

## 4 Experiments

In this section we present our experiments on optical character recognition. We observed the following. First, the optimal convex combination of kernels computed by our algorithm is competitive with the best base kernels. Second, by observing the 'weights' of the convex combination we can distinguish the strong from the weak candidate kernels. We proceed by discussing the details of the experimental design interleaved with our results.

We used the USPS dataset[2] of $16 \times 16$ images of handwritten digits with pixel values ranging between -1 and 1. We present the results for 5 pairwise classification tasks of varying difficulty and for odd vs. even digit classification. For pairwise classification, the training set consisted of the first 200 images for each digit in the USPS training set and the number of labeled points was chosen to be 4, 8 or 12 (with equal numbers for each digit). For odd vs. even digit classification, the training set consisted of the first 80 images per digit in the USPS training set and the number of labeled points was 10, 20 or 30, with equal numbers for each digit. Performance was averaged over 30 random selections, each with the same number of labeled points.

In each experiment, we constructed $n = 30$ graphs $G^{(q)}$ ($q \in \mathbb{N}_n$) by combining $k$-nearest neighbors ($k \in \mathbb{N}_{10}$) with three different distances. Then, $n$ corresponding Laplacians were computed together with their associated kernels. We chose as the loss function $V$ the square loss. Since kernels obtained from different types of graphs can vary widely, it was necessary to renormalize them. Hence, we chose to normalize each kernel during the

[2]Available at: *http://www-stat-class.stanford.edu/~tibs/ElemStatLearn/data.html*

| Task \ Labels % | Euclidean (10 kernels) | | | Transf. (10 kernels) | | | Tangent dist. (10 kernels) | | | All (30 kernels) | | |
|---|---|---|---|---|---|---|---|---|---|---|---|---|
| | 1% | 2% | 3% | 1% | 2% | 3% | 1% | 2% | 3% | 1% | 2% | 3% |
| 1 vs. 7 | 1.55 | 1.53 | 1.50 | 1.45 | 1.45 | 1.38 | 1.01 | 1.00 | 1.00 | 1.28 | 1.24 | 1.20 |
| | 0.08 | 0.05 | 0.15 | 0.10 | 0.11 | 0.12 | 0.00 | 0.09 | 0.11 | 0.28 | 0.27 | 0.22 |
| 2 vs. 3 | 3.08 | 3.34 | 3.38 | 0.80 | 0.85 | 0.82 | 0.73 | 0.19 | 0.03 | 0.79 | 0.25 | 0.10 |
| | 0.85 | 1.21 | 1.29 | 0.40 | 0.38 | 0.32 | 0.93 | 0.51 | 0.09 | 0.93 | 0.61 | 0.21 |
| 2 vs. 7 | 4.46 | 4.04 | 3.56 | 3.27 | 2.92 | 2.96 | 2.95 | 2.30 | 2.14 | 3.51 | 2.54 | 2.41 |
| | 1.17 | 1.21 | 0.82 | 1.16 | 1.26 | 1.08 | 1.79 | 0.76 | 0.53 | 1.92 | 0.97 | 0.89 |
| 3 vs. 8 | 7.33 | 7.30 | 7.03 | 6.98 | 6.87 | 6.50 | 4.43 | 4.22 | 3.96 | 4.80 | 4.32 | 4.20 |
| | 1.67 | 1.49 | 1.43 | 1.57 | 1.77 | 1.78 | 1.21 | 1.36 | 1.25 | 1.57 | 1.46 | 1.53 |
| 4 vs. 7 | 2.90 | 2.64 | 2.25 | 1.81 | 1.82 | 1.69 | 0.88 | 0.90 | 0.90 | 1.04 | 1.14 | 1.13 |
| | 0.77 | 0.78 | 0.77 | 0.26 | 0.42 | 0.45 | 0.17 | 0.20 | 0.20 | 0.37 | 0.42 | 0.39 |
| Labels | 10 | 20 | 30 | 10 | 20 | 30 | 10 | 20 | 30 | 10 | 20 | 30 |
| Odd vs. Even | 18.6 | 15.5 | 13.4 | 15.7 | 11.7 | 8.52 | 14.66 | 10.50 | 8.38 | 17.07 | 10.98 | 8.74 |
| | 3.98 | 2.40 | 2.67 | 4.40 | 3.14 | 1.32 | 4.37 | 2.30 | 1.90 | 4.38 | 2.61 | 2.39 |

Table 1: Misclassification error percentage (*top*) and standard deviation (*bottom*) for the best convex combination of kernels on different handwritten digit recognition tasks, using different distances. See text for description.

training process by the Frobenius norm of its submatrix corresponding to the labeled data. We also observed that similar results were obtained when normalizing with the trace of this submatrix. The regularization parameter was set to $10^{-5}$ in all algorithms. For convex minimization, as the starting kernel in the algorithm in Figure 1 we always used the average of the $n$ kernels and as the maximum number of iterations $T = 100$.

Table 1 shows the results obtained using three distances as combined with $k$-NN ($k \in \mathbb{N}_{10}$). The first distance is the *Euclidean* distance between images. The second method is *transformation*, where the distance between two images is given by the smallest Euclidean distance between any pair of transformed images as determined by applying a number of affine transformations and a thickness transformation[3], see [6] for more information. The third distance is *tangent distance*, as described in [6], which is a first-order approximation to the above transformations. For the first three columns in the table the Euclidean distance was used, for columns 4–6 the image transformation distance was used, for columns 7–9 the tangent distance was used. Finally, in the last three columns all three methods were jointly compared. As the results indicate, when combining different types of kernels, the algorithm tends to select the most effective ones (in this case the tangent distance kernels and to a lesser degree the transformation distance kernels which did not work very well because of the Matlab optimization routine we used). We also noted that within each of the methods the performance of the convex combination is comparable to that of the best kernels. Figure 2 reports the weight of each individual kernel learned by our algorithm when 2% labels are used in the pairwise tasks and 20 labels are used for odd vs. even. With the exception of the easy 1 vs. 7 task, the large weights are associated with the graphs/kernels built with the tangent distance.

The effectiveness of our algorithm in selecting the good graphs/kernels is better demonstrated in Figure 3, where the Euclidean and the transformation kernels are combined with a "low-quality" kernel. This "low-quality" kernel is induced by considering distances invariant over rotation in the range $[-180°, 180°]$, so that the image of a 6 can easily have a small distance from an image of a 9, that is, if $\mathbf{x}$ and $\mathbf{t}$ are two images and $T_\theta(\mathbf{x})$ is the image obtained by rotating $\mathbf{x}$ by $\theta$ degrees, we set

$$d(\mathbf{x}, \mathbf{t}) = \min\{\|T_\theta(\mathbf{x}) - T_{\theta'}(\mathbf{t})\| : \theta, \theta' \in [-180°, 180°]\}.$$

The figure shows the distance matrix on the set of labeled and unlabeled data for the Euclidean, transformation and "low-quality distance" respectively. The best error among 15 different values of $k$ within each method, the error of the learned convex combination and the total learned weights for each method are shown below each plot. It is clear that the solution of the algorithm is dominated by the good kernels and is not influenced by the ones with low performance. As a result, the error of the convex combination is comparable to that of the Euclidean and transformation methods. The final experiment (see Figure 4) demonstrates that unlabeled data improves the performance of our method.

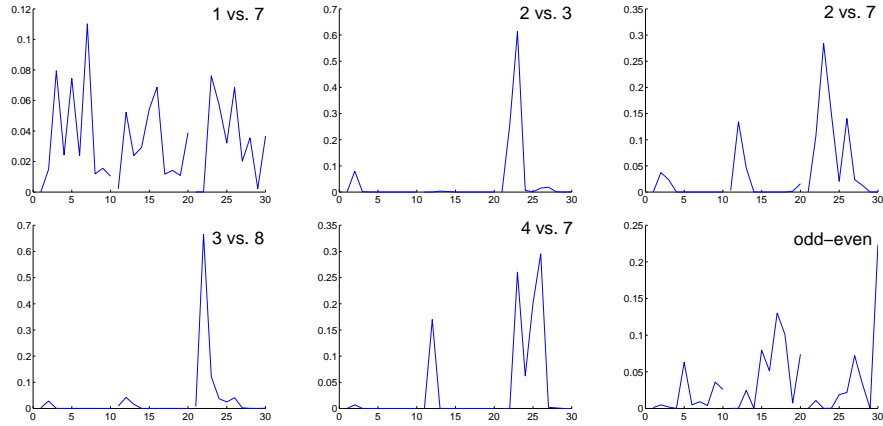

Figure 2: Kernel weights for Euclidean (first 10), Transformation (middle 10) and Tangent (last 10). See text for more information.

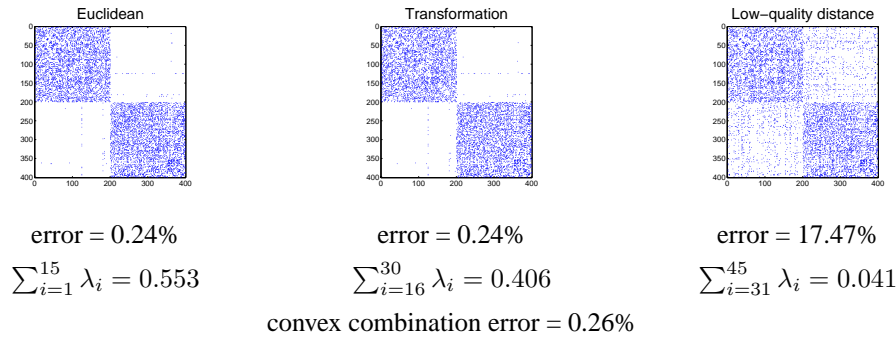

| error = 0.24% | error = 0.24% | error = 17.47% |
|---|---|---|
| $\sum_{i=1}^{15} \lambda_i = 0.553$ | $\sum_{i=16}^{30} \lambda_i = 0.406$ | $\sum_{i=31}^{45} \lambda_i = 0.041$ |

convex combination error = 0.26%

Figure 3: Similarity matrices and corresponding learned coefficients of the convex combination for the 6 vs. 9 task. See text for description.

## 5 Conclusion

We have presented a method for computing an optimal kernel within the framework of regularization over graphs. The method consists of a minimax problem which can be efficiently solved by using an algorithm from [1]. When tested on optical character recognition tasks, the method exhibits competitive performance and is able to select good graph structures. Future work will focus on out-of-sample extensions of this algorithm and on continuous optimization versions of it. In particular, we may consider a continuous family of graphs each corresponding to a different weight matrix and study graph kernel combinations over this class.

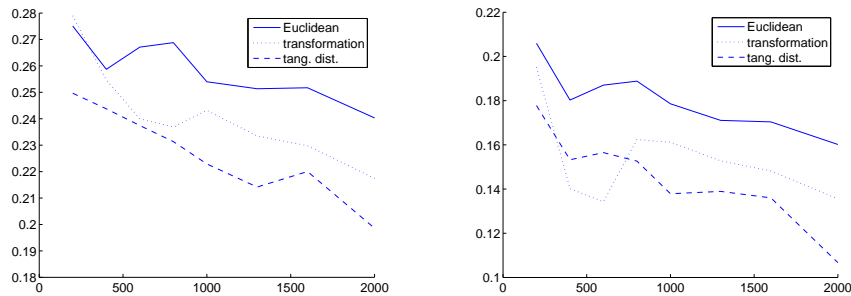

Figure 4: Misclassification error vs. number of training points for odd vs. even classification. The number of labeled points is 10 on the left and 20 on the right.

## Footnotes

[1] The ideas we discuss below naturally extend to weighted graphs.

[3]This distance was approximated using Matlab's constrained minimization function.

# References

[1] A. Argyriou, C.A. Micchelli and M. Pontil. Learning convex combinations of continuously parameterized basic kernels. Proc. 18-th Conf. on Learning Theory, 2005.

[2] M. Belkin, I. Matveeva and P. Niyogi. Regularization and semi-supervised learning on large graphs. Proc. of 17–th Conf. Learning Theory (COLT), 2004.

[3] M. Belkin and P. Niyogi. Semi-supervised learning on Riemannian manifolds. *Mach. Learn.*, 56: 209–239, 2004.

[4] A. Blum and S. Chawla. Learning from Labeled and Unlabeled Data using Graph Mincuts, Proc. of 18–th International Conf. on Learning Theory, 2001.

[5] F.R. Chung. *Spectral Graph Theory*. Regional Conference Series in Mathematics, Vol. 92, 1997.

[6] T. Hastie and P. Simard. Models and Metrics for Handwritten Character Recognition. *Statistical Science*, 13(1): 54–65, 1998.

[7] M. Herbster, M. Pontil, L. Wainer. Online learning over graphs. Proc. 22-nd Int. Conf. Machine Learning, 2005.

[8] T. Joachims. Transductive Learning via Spectral Graph Partitioning. Proc. of the Int. Conf. Machine Learning (ICML), 2003.

[9] R.I. Kondor and J. Lafferty. Diffusion kernels on graphs and other discrete input spaces. Proc. 19-th Int. Conf. Machine Learning, 2002.

[10] G. R. G. Lanckriet, N. Cristianini, P. Bartlett, L. El Ghaoui, M. I. Jordan. Learning the kernel matrix with semidefinite programming. *J. Machine Learning Research,* 5: 27–72, 2004.

[11] Y. Lin and H.H. Zhang. Component selection and smoothing in smoothing spline analysis of variance models – COSSO. Institute of Statistics Mimeo Series 2556, NCSU, January 2003.

[12] C. A. Micchelli and M. Pontil. Learning the kernel function via regularization, *J. Machine Learning Research*, 6: 1099–1125, 2005.

[13] C.S. Ong, A.J. Smola, and R.C. Williamson. Hyperkernels. *Advances in Neural Information Processing Systems*, 15, S. Becker et al. (Eds.), MIT Press, Cambridge, MA, 2003.

[14] A.J. Smola and R.I Kondor. Kernels and regularization on graphs. Proc. of 16–th Conf. Learning Theory (COLT), 2003.

[15] V.N. Vapnik. *Statistical Learning Theory*. Wiley, New York, 1998.

[16] D. Zhou, O. Bousquet, T.N. Lal, J. Weston and B. Scholkopf. Learning with local and global consistency. *Advances in Neural Information Processing Systems*, 16, S. Thrun et al. (Eds.), MIT Press, Cambridge, MA, 2004.

[17] X. Zhu, Z. Ghahramani and J. Lafferty. Semi-supervised learning using Gaussian fields and harmonic functions. Proc. 20–th Int. Conf. Machine Learning, 2003.

[18] X. Zhu, J. Kandola, Z, Ghahramani, J. Lafferty. Nonparametric transforms of graph kernels for semi-supervised learning. *Advances in Neural Information Processing Systems*, 17, L.K. Saul et al. (Eds.), MIT Press, Cambridge, MA, 2005.
